# Minimum Bayes Error Feature Selection for Continuous Speech Recognition

**George Saon and Mukund Padmanabhan**
IBM T. J. Watson Research Center, Yorktown Heights, NY, 10598
E-mail: {saon,mukund}@watson.ibm.com, Phone: (914)-945-2985

## Abstract

We consider the problem of designing a linear transformation $\theta \in \mathbb{R}^{p \times n}$, of rank $p \leq n$, which projects the features of a classifier $\mathbf{x} \in \mathbb{R}^n$ onto $\mathbf{y} = \theta \mathbf{x} \in \mathbb{R}^p$ such as to achieve minimum Bayes error (or probability of misclassification). Two avenues will be explored: the first is to maximize the $\theta$-average divergence between the class densities and the second is to minimize the union Bhattacharyya bound in the range of $\theta$. While both approaches yield similar performance in practice, they outperform standard LDA features and show a 10% relative improvement in the word error rate over state-of-the-art cepstral features on a large vocabulary telephony speech recognition task.

## 1 Introduction

Modern speech recognition systems use cepstral features characterizing the short-term spectrum of the speech signal for classifying frames into phonetic classes. These features are augmented with dynamic information from the adjacent frames to capture transient spectral events in the signal. What is commonly referred to as MFCC+$\Delta + \Delta\Delta$ features consist in "static" mel-frequency cepstral coefficients (usually 13) plus their first and second order derivatives computed over a sliding window of typically 9 consecutive frames yielding 39-dimensional feature vectors every 10ms. One major drawback of this front-end scheme is that the same computation is performed regardless of the application, channel conditions, speaker variability, etc. In recent years, an alternative feature extraction procedure based on discriminant techniques has emerged: the consecutive cepstral frames are spliced together forming a supervector which is then projected down to a manageable dimension. One of the most popular objective functions for designing the feature space projection is linear discriminant analysis.

LDA [2, 3] is a standard technique in statistical pattern classification for dimensionality reduction with a minimal loss in discrimination. Its application to speech recognition has shown consistent gains for small vocabulary tasks and mixed results for large vocabulary applications [4, 6]. Recently, there has been an interest in extending LDA to heteroscedastic discriminant analysis (HDA) by incorporating the individual class covariances in the objective function [6, 8]. Indeed, the equal class covariance assumption made by LDA does

not always hold true in practice making the LDA solution highly suboptimal for specific cases [8].

However, since both LDA and HDA are heuristics, they do not guarantee an optimal projection in the sense of a minimum Bayes classification error. The aim of this paper is to study feature space projections according to objective functions which are more intimately linked to the probability of misclassification. More specifically, we will define the probability of misclassification in the original space, $\epsilon$, and in the projected space, $\epsilon_\theta$, and give conditions under which $\epsilon_\theta = \epsilon$. Since after a projection $\mathbf{y} = \theta\mathbf{x}$ discrimination information is usually lost, the Bayes error in the projected space will always increase, that is $\epsilon_\theta \geq \epsilon$ therefore minimizing $\epsilon_\theta$ amounts to finding $\theta$ for which the equality case holds. An alternative approach is to define an upper bound on $\epsilon_\theta$ and to directly minimize this bound.

The paper is organized as follows: in section 2 we recall the definition of the Bayes error rate and its link to the divergence and the Bhattacharyya bound, section 3 deals with the experiments and results and section 4 provides a final discussion.

## 2 Bayes error, divergence and Bhattacharyya bound

### 2.1 Bayes error

Consider the general problem of classifying an $n$-dimensional vector $\mathbf{x}$ into one of $C$ distinct classes. Let each class $i$ be characterized by its own prior $\lambda_i$ and probability density function $p_i, i = 1, \ldots, C$. Suppose $\mathbf{x}$ is classified as belonging to class $j$ through the Bayes assignment $j = \text{argmax}_{1 \leq i \leq C} \lambda_i p_i(\mathbf{x})$. The expected error rate for this classifier is called Bayes error [3] or probability of misclassification and is defined as

$$\epsilon = 1 - \int_{\mathbf{R}^n} \max_{1 \leq i \leq C} \lambda_i p_i(\mathbf{x}) d\mathbf{x} \tag{1}$$

Suppose next that we wish to perform the linear transformation $f : \mathbf{R}^n \to \mathbf{R}^p$, $\mathbf{y} = f(\mathbf{x}) = \theta\mathbf{x}$, with $\theta$ a $p \times n$ matrix of rank $p \leq n$. Moreover, let us denote by $p_i^\theta$ the transformed density for class $i$. The Bayes error in the range of $\theta$ now becomes

$$\epsilon_\theta = 1 - \int_{\mathbf{R}^p} \max_{1 \leq i \leq C} \lambda_i p_i^\theta(\mathbf{y}) d\mathbf{y} \tag{2}$$

Since the transformation $\mathbf{y} = \theta\mathbf{x}$ produces a vector whose coefficients are linear combinations of the input vector $\mathbf{x}$, it can be shown [1] that, in general, information is lost and $\epsilon_\theta \geq \epsilon$.

For a fixed $p$, the feature selection problem can be stated as finding $\hat{\theta}$ such that

$$\hat{\theta} = \underset{\theta \in \mathbf{R}^{p \times n}, \ rank(\theta) = p}{\text{argmin}} \epsilon_\theta \tag{3}$$

We will take however an indirect approach to (3): by maximizing the average pairwise divergence and relating it to $\epsilon_\theta$ (subsection 2.2) and by minimizing the union Bhattacharyya bound on $\epsilon_\theta$ (subsection 2.3).

## 2.2 Interclass divergence

Since Kullback [5], the symmetric divergence between class $i$ and $j$ is given by

$$D(i,j) = \int_{\mathbf{R}^n} p_i(\mathbf{x}) \log \frac{p_i(\mathbf{x})}{p_j(\mathbf{x})} + p_j(\mathbf{x}) \log \frac{p_j(\mathbf{x})}{p_i(\mathbf{x})} d\mathbf{x} \qquad (4)$$

$D(i,j)$ represents a measure of the degree of difficulty of discriminating between the classes (the larger the divergence, the greater the separability between the classes). Similarly, one can define $D_\theta(i,j)$, the pairwise divergence in the range of $\theta$. Kullback [5] showed that $D_\theta(i,j) \leq D(i,j)$. If the equality case holds, $\theta$ is called a *sufficient statistic for discrimination*. The average pairwise divergence is defined as $D = \frac{2}{C(C-1)} \sum_{1 \leq i < j \leq C} D(i,j)$ and respectively $D_\theta = \frac{2}{C(C-1)} \sum_{1 \leq i < j \leq C} D_\theta(i,j)$. It follows that $D_\theta \leq D$. The next theorem due to Decell [1] provides a link between Bayes error and divergence for classes with uniform priors $\lambda_1 = \ldots = \lambda_C (= \frac{1}{C})$.

**Theorem [Decell'72]** *If $D_\theta = D$ then $\epsilon_\theta = \epsilon$.*

The main idea of the proof is to show that if the divergences are the same then the Bayes assignment is preserved because the likelihood ratios are preserved almost everywhere: $\frac{p_i(\mathbf{x})}{p_j(\mathbf{x})} = \frac{p_i^\theta(\theta\mathbf{x})}{p_j^\theta(\theta\mathbf{x})}$, $i \neq j$. The result follows by noting that for any measurable set $A \subset \mathbb{R}^p$

$$\int_A p_i^\theta(\mathbf{y}) d\mathbf{y} = \int_{\theta^{-1}(A)} p_i(\mathbf{x}) d\mathbf{x} \qquad (5)$$

where $\theta^{-1}(A) = \{\mathbf{x} \in \mathbb{R}^n | \theta\mathbf{x} \in A\}$. The previous theorem provides a basis for selecting $\theta$ such as to maximize $D_\theta$.

Let us make next the assumption that each class $i$ is normally distributed with mean $\mu_i$ and covariance $\Sigma_i$, that is $p_i(\mathbf{x}) = \mathcal{N}(\mathbf{x}; \mu_i, \Sigma_i)$ and $p_i^\theta(\mathbf{y}) = \mathcal{N}(\mathbf{y}; \theta\mu_i, \theta\Sigma_i\theta^T)$, $i = 1, \ldots, C$. It is straightforward to show that in this case the divergence is given by

$$D(i,j) = \frac{1}{2} \operatorname{trace}\{\Sigma_i^{-1}[\Sigma_j + (\mu_i - \mu_j)(\mu_i - \mu_j)^T] + \Sigma_j^{-1}[\Sigma_i + (\mu_i - \mu_j)(\mu_i - \mu_j)^T]\} - n$$
$$(6)$$

Thus, the objective function to be maximized becomes

$$D_\theta = \frac{1}{C(C-1)} \operatorname{trace}\{\sum_{i=1}^{C} (\theta\Sigma_i\theta^T)^{-1}\theta S_i\theta^T\} - p \qquad (7)$$

where $S_i = \sum_{j \neq i} \Sigma_j + (\mu_i - \mu_j)(\mu_i - \mu_j)^T$, $i = 1, \ldots, C$.

Following matrix differentiation results from [9], the gradient of $D_\theta$ with respect to $\theta$ has the expression

$$\frac{\partial D_\theta}{\partial \theta} = \frac{1}{C(C-1)} \sum_{i=1}^{C} (\theta\Sigma_i\theta^T)^{-1}[\theta S_i\theta^T (\theta\Sigma_i\theta^T)^{-1}\theta\Sigma_i - \theta S_i] \qquad (8)$$

Unfortunately, it turns out that $\frac{\partial D_\theta}{\partial \theta} = 0$ has no analytical solutions for the stationary points. Instead, one has to use numerical optimization routines for the maximization of $D_\theta$.

## 2.3 Bhattacharyya bound

An alternative way of minimizing the Bayes error is to minimize an upper bound on this quantity. We will first prove the following statement

$$\epsilon \leq \sum_{1 \leq i < j \leq C} \sqrt{\lambda_i \lambda_j} \int_{\mathbb{R}^n} \sqrt{p_i(\mathbf{x}) p_j(\mathbf{x})} d\mathbf{x} \qquad (9)$$

Indeed, from (1), the Bayes error can be rewritten as

$$
\begin{aligned}
\epsilon &= \int_{\mathbb{R}^n} \sum_{i=1}^{C} \lambda_i p_i(\mathbf{x}) d\mathbf{x} - \int_{\mathbb{R}^n} \max_{1 \leq i \leq C} \lambda_i p_i(\mathbf{x}) d\mathbf{x} \\
&= \int_{\mathbb{R}^n} \min_{1 \leq i \leq C} \sum_{j \neq i} \lambda_j p_j(\mathbf{x}) d\mathbf{x}
\end{aligned}
\qquad (10)
$$

and for every $\mathbf{x}$, there exists a permutation of the indices $\sigma_\mathbf{x} : \{1, \ldots, C\} \to \{1, \ldots, C\}$ such that the terms $\lambda_1 p_1(\mathbf{x}), \ldots, \lambda_C p_C(\mathbf{x})$ are sorted in increasing order, i.e. $\lambda_{\sigma_\mathbf{x}(1)} p_{\sigma_\mathbf{x}(1)}(\mathbf{x}) \leq \ldots \leq \lambda_{\sigma_\mathbf{x}(C)} p_{\sigma_\mathbf{x}(C)}(\mathbf{x})$. Moreover, for $1 \leq k \leq C - 1$

$$\lambda_{\sigma_\mathbf{x}(k)} p_{\sigma_\mathbf{x}(k)}(\mathbf{x}) \leq \sqrt{\lambda_{\sigma_\mathbf{x}(k)} p_{\sigma_\mathbf{x}(k)}(\mathbf{x}) \lambda_{\sigma_\mathbf{x}(k+1)} p_{\sigma_\mathbf{x}(k+1)}(\mathbf{x})} \qquad (11)$$

from which follows that

$$\min_{1 \leq i \leq C} \sum_{j \neq i} \lambda_j p_j(\mathbf{x}) = \sum_{k=1}^{C-1} \lambda_{\sigma_\mathbf{x}(k)} p_{\sigma_\mathbf{x}(k)}(\mathbf{x}) \leq \sum_{k=1}^{C-1} \sqrt{\lambda_{\sigma_\mathbf{x}(k)} p_{\sigma_\mathbf{x}(k)}(\mathbf{x}) \lambda_{\sigma_\mathbf{x}(k+1)} p_{\sigma_\mathbf{x}(k+1)}(\mathbf{x})}$$

$$\leq \sum_{1 \leq i < j \leq C} \sqrt{\lambda_i p_i(\mathbf{x}) \lambda_j p_j(\mathbf{x})}$$

$$(12)$$

which, when integrated over $\mathbb{R}^n$, leads to (9).

As previously, if we assume that the $p_i$'s are normal distributions with means $\mu_i$ and covariances $\Sigma_i$, the bound given by the right-hand side of (9) has the closed form expression

$$\sum_{1 \leq i < j \leq C} \sqrt{\lambda_i \lambda_j} e^{-\rho(i,j)} \qquad (13)$$

where

$$\rho(i,j) = \frac{1}{8} (\mu_i - \mu_j)^T \left[ \frac{\Sigma_i + \Sigma_j}{2} \right]^{-1} (\mu_i - \mu_j) + \frac{1}{2} \log \frac{\left| \frac{\Sigma_i + \Sigma_j}{2} \right|}{\sqrt{|\Sigma_i||\Sigma_j|}} \qquad (14)$$

is called the Bhattacharyya distance between the normal distributions $p_i$ and $p_j$ [3]. Similarly, one can define $\rho_\theta(i,j)$, the Bhattacharyya distance between the projected densities $p_i^\theta$ and $p_j^\theta$. Combining (9) and (13), one obtains the following inequality involving the Bayes error rate in the projected space

$$\epsilon_\theta \leq \sum_{1 \leq i < j \leq C} \sqrt{\lambda_i \lambda_j} e^{-\rho_\theta(i,j)} (= B_\theta) \tag{15}$$

It is necessary at this point to introduce the following simplifying notations:

- $B_{ij} = \frac{1}{4}(\mu_i - \mu_j)(\mu_i - \mu_j)^T$ and
- $W_{ij} = \frac{1}{2}(\Sigma_i + \Sigma_j), 1 \leq i < j \leq C.$

From (14), it follows that

$$\rho_\theta(i,j) = \frac{1}{2} \operatorname{trace}\{(\theta W_{ij}\theta^T)^{-1}\theta B_{ij}\theta^T\} + \frac{1}{2} \log \frac{|\theta W_{ij}\theta^T|}{\sqrt{|\theta \Sigma_i \theta^T||\theta \Sigma_j \theta^T|}} \tag{16}$$

and the gradient of $B_\theta$ with respect to $\theta$ is

$$\frac{\partial B_\theta}{\partial \theta} = - \sum_{1 \leq i < j \leq C} \sqrt{\lambda_i \lambda_j} e^{-\rho_\theta(i,j)} \frac{\partial \rho_\theta(i,j)}{\partial \theta} \tag{17}$$

with, again by making use of differentiation results from [9]

$$\begin{aligned}
\frac{\partial \rho_\theta(i,j)}{\partial \theta} &= \frac{1}{2}(\theta W_{ij}\theta^T)^{-1}[\theta B_{ij}\theta^T(\theta W_{ij}\theta^T)^{-1}\theta W_{ij} - \theta B_{ij}] + (\theta W_{ij}\theta^T)^{-1}\theta W_{ij} - \\
&\quad - \frac{1}{2}[(\theta \Sigma_i \theta^T)^{-1}\theta \Sigma_i + (\theta \Sigma_j \theta^T)^{-1}\theta \Sigma_j]
\end{aligned} \tag{18}$$

## 3  Experiments and results

The speech recognition experiments were conducted on a voicemail transcription task [7]. The baseline system has 2.3K context dependent HMM states and 134K diagonal gaussian mixture components and was trained on approximately 70 hours of data. The test set consists of 86 messages (approximately 7000 words). The baseline system uses 39-dimensional frames (13 cepstral coefficients plus deltas and double deltas computed from 9 consecutive frames). For the divergence and Bhattacharyya projections, every 9 consecutive 24-dimensional cepstral vectors were spliced together forming 216-dimensional feature vectors which were then clustered to estimate 1 full covariance gaussian density for each state. Subsequently, a 39×216 transformation $\theta$ was computed using the objective functions for the divergence (7) and the Bhattacharyya bound (15), which projected the models and feature space down to 39 dimensions. As mentioned in [4], it is not clear what the most appropriate class definition for the projections should be. The best results were obtained by considering each individual HMM state as a separate class, with the priors of the gaussians summing up to one across states. Both optimizations were initialized with

the LDA matrix and carried out using a conjugate gradient descent routine with user supplied analytic gradient from the NAG[1] Fortran library. The routine performs an iterative update of the inverse of the hessian of the objective function by accumulating curvature information during the optimization.

Figure 1 shows the evolution of the objective functions for the divergence and the Bhattacharyya bound.

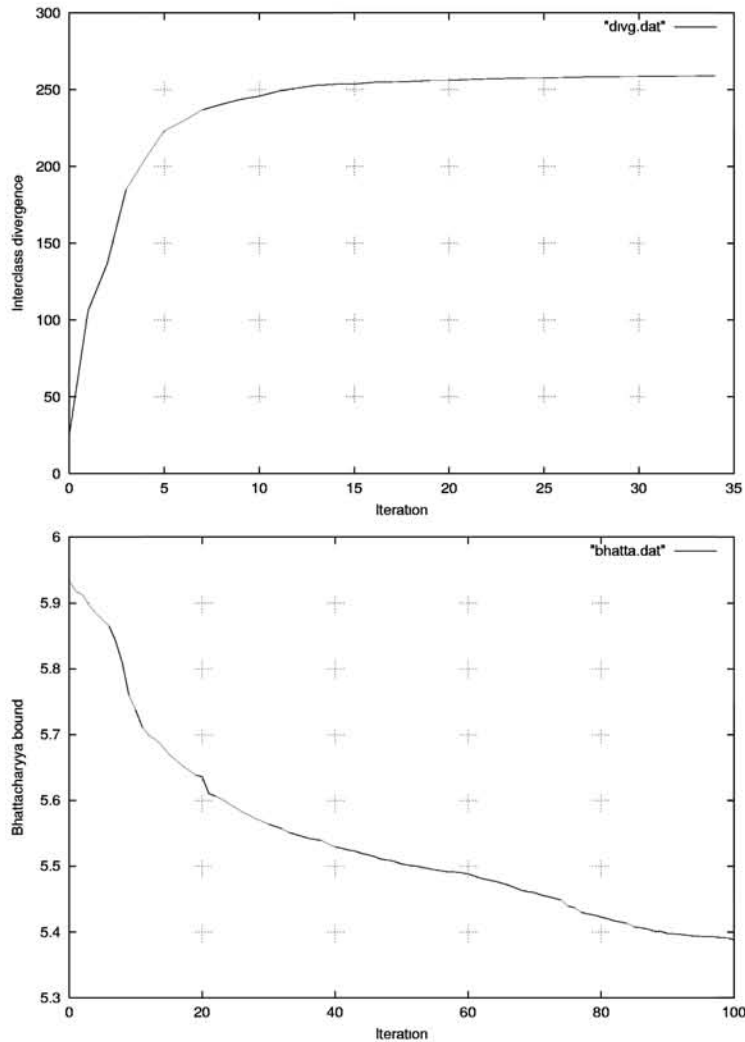

Figure 1: Evolution of the objective functions.

The parameters of the baseline system (with 134K gaussians) were then re-estimated in the transformed spaces using the EM algorithm. Table 1 summarizes the improvements in the word error rates for the different systems.

| System | Word error rate |
|---|---|
| Baseline (MFCC+$\Delta + \Delta\Delta$) | 39.61% |
| LDA | 37.39% |
| Interclass divergence | 36.32% |
| Bhattacharyya bound | 35.73% |

Table 1: Word error rates for the different systems.

## 4 Summary

Two methods for performing discriminant feature space projections have been presented. Unlike LDA, they both aim to minimize the probability of misclassification in the projected space by either maximizing the interclass divergence and relating it to the Bayes error or by directly minimizing an upper bound on the classification error. Both methods lead to defining smooth objective functions which have as argument projection matrices and which can be numerically optimized. Experimental results on large vocabulary continuous speech recognition over the telephone show the superiority of the resulting features over their LDA or cepstral counterparts.

## Footnotes

[1]Numerical Algebra Group

## References

[1] H. P. Decell and J. A. Quirein. An iterative approach to the feature selection problem. *Proc. Purdue Univ. Conf. on Machine Processing of Remotely Sensed Data*, 3B1-3B12, 1972.

[2] R. O. Duda and P. B. Hart. Pattern classification and scene analysis. *Wiley*, New York, 1973.

[3] K. Fukunaga. Introduction to statistical pattern recognition. *Academic Press*, New York, 1990.

[4] R. Haeb-Umbach and H. Ney. Linear Discriminant Analysis for improved large vocabulary continuous speech recognition. *Proceedings of ICASSP'92*, volume 1, pages 13–16, 1992.

[5] S. Kullback. Information theory and statistics. *Wiley*, New York, 1968.

[6] N. Kumar and A. G. Andreou. Heteroscedastic discriminant analysis and reduced rank HMMs for improved speech recognition. *Speech Communcation*, 26:283–297, 1998.

[7] M. Padmanabhan, G. Saon, S. Basu, J. Huang and G. Zweig. Recent improvements in voicemail transcription. *Proceedings of EUROSPEECH'99*, Budapest, Hungary, 1999.

[8] G. Saon, M. Padmanabhan, R. Gopinath and S. Chen. Maximum likelihood discriminant feature spaces. *Proceedings of ICASSP'2000*, Istanbul, Turkey, 2000.

[9] S. R. Searle. Matrix algebra useful for statistics. *Wiley Series in Probability and Mathematical Statistics*, New York, 1982.
